# Online Learning with Kernels

**Jyrki Kivinen**    **Alex J. Smola**    **Robert C. Williamson**
Research School of Information Sciences and Engineering
Australian National University
Canberra, ACT 0200

## Abstract

We consider online learning in a Reproducing Kernel Hilbert Space. Our method is computationally efficient and leads to simple algorithms. In particular we derive update equations for classification, regression, and novelty detection. The inclusion of the $\nu$-trick allows us to give a robust parameterization. Moreover, unlike in batch learning where the $\nu$-trick only applies to the $\varepsilon$-insensitive loss function we are able to derive general trimmed-mean types of estimators such as for Huber's robust loss.

## 1   Introduction

While kernel methods have proven to be successful in many batch settings (Support Vector Machines, Gaussian Processes, Regularization Networks) the extension to online methods has proven to provide some unsolved challenges. Firstly, the standard online settings for linear methods are in danger of overfitting, when applied to an estimator using a feature space method. This calls for regularization (or prior probabilities in function space if the Gaussian Process view is taken).

Secondly, the functional representation of the estimator becomes more complex as the number of observations increases. More specifically, the Representer Theorem [10] implies that the number of kernel functions can grow up to linearly with the number of observations. Depending on the loss function used [15], this *will* happen in practice in most cases. Thereby the complexity of the estimator used in prediction increases linearly over time (in some restricted situations this can be reduced to logarithmic cost [8]).

Finally, training time of batch and/or incremental update algorithms typically increases superlinearly with the number of observations. Incremental update algorithms [2] attempt to overcome this problem but cannot guarantee a bound on the number of operations required per iteration. Projection methods [3] on the other hand, will ensure a limited number of updates per iteration. However they can be computationally expensive since they require one matrix multiplication at each step. The size of the matrix is given by the number of kernel functions required at each step.

Recently several algorithms have been proposed [5, 8, 6, 12] performing perceptron-like updates for classification at each step. Some algorithms work only in the noise free case, others not for moving targets, and yet again others assume an upper bound on the complexity of the estimators. In the present paper we present a simple method which will allows the use of kernel estimators for classification, regression, and novelty detection and which copes with a large number of kernel functions efficiently.

## 2 Stochastic Gradient Descent in Feature Space

**Reproducing Kernel Hilbert Space**   The class of functions $f : \mathcal{X} \to \mathbb{R}$ to be studied in this paper are elements of an RKHS $\mathcal{H}$. This means that there exists a kernel $k : \mathcal{X} \times \mathcal{X} \to \mathbb{R}$ and a dot product $\langle \cdot, \cdot \rangle$ such that 1) $\langle f(\cdot), k(x, \cdot) \rangle = f(x)$ (reproducing property); 2) $\mathcal{H}$ is the closure of the span of all $k(x, \cdot)$ with $x \in \mathcal{X}$. In other words, all $f \in \mathcal{H}$ are linear combinations of kernel functions.

Typically $\|f\|^2 = \langle f, f \rangle$ is used as a regularization functional. It is the "length of the weight vector in feature space" as commonly used in SV algorithms. To state our algorithm we need to compute derivatives of functionals defined on $\mathcal{H}$.

For the regularizer $\Omega[f] := \frac{1}{2}\|f\|^2$ we obtain $\partial_f \Omega[f] = f$. More general versions of $\Omega[f] = \omega(\|f\|)$ lead to $\partial_f \Omega[f] = \omega'(\|f\|)\|f\|^{-1}f$.

For the evaluation functional $e_x[f] := f(x)$ we compute the derivative by using the reproducing property of $\mathcal{H}$ and obtain $\partial_f e_x[f] = \partial_f \langle f(\cdot), k(x, \cdot) \rangle = k(x, \cdot)$. Consequently for a function $c : \mathcal{X} \times \mathcal{Y} \times \mathcal{Y} \to \mathbb{R}$ which is differentiable in its third argument we obtain $\partial_f c(x, y, f(x)) = c'(x, y, f(x))k(x, \cdot)$. Below $c$ will be the loss function.

**Regularized Risk Functionals and Learning**   In the standard learning setting we are supplied with pairs of observations $(x_i, y_i) \in \mathcal{X} \times \mathcal{Y}$ drawn according to some underlying distribution $P(x, y)$. Our aim is to predict the likely outcome $y$ at location $x$. Several variants are possible: (i) $P(x, y)$ may change over time, (ii) the training sample $(x_i, y_i)$ may be the next observation on which to predict which leads to a true online setting, or (iii) we may want to find an algorithm which approximately minimizes a regularized risk functional on a given training set.

We assume that we want to minimize a loss function $c : \mathcal{X} \times \mathcal{Y} \times \mathcal{Y} \to \mathbb{R}$ which penalizes the deviation between an observation $y$ at location $x$ and the prediction $f(x)$, based on observations $(x_1, y_1), \ldots, (x_m, y_m)$. Since $P(x, y)$ is unknown, a standard approach is to instead minimize the empirical risk

$$R_{\text{emp}}[f] = \frac{1}{m}\sum_{i=1}^{m} c(x_i, y_i, f(x_i)) \tag{1}$$

or, in order to avoid overly complex hypotheses, minimize the empirical risk plus an additional regularization term $\Omega[f]$. This sum is known as the regularized risk

$$R_{\text{reg}}[f] := R_{\text{emp}}[f] + \lambda\Omega[f] = \frac{1}{m}\sum_{i=1}^{m} c(x_i, y_i, f(x_i)) + \lambda\Omega[f] \text{ for } \lambda > 0. \tag{2}$$

Common loss functions are the soft margin loss function [1] or the logistic loss for classification and novelty detection [14], the quadratic loss, absolute loss, Huber's robust loss [9], or the $\varepsilon$-insensitive loss [16] for regression. We discuss these in Section 3.

In some cases the loss function depends on an additional parameter such as the width of the margin $\rho$ or the size of the $\varepsilon$-insensitive zone. One may make these variables themselves parameters of the optimization problem [15] in order to make the loss function adaptive to the amount or type of noise present in the data. This typically results in a term $\nu\varepsilon$ or $-\nu\rho$ added to $c(x, y, f(x))$.

**Stochastic Approximation**   In order to find a good estimator we would like to minimize $R_{\text{reg}}[f]$. This can be costly if the number of observations is large. Recently several gradient descent algorithms for minimizing such functionals efficiently have been proposed [13, 7]. Below we extend these methods to *stochastic* gradient descent by approximating $R_{\text{reg}}[f]$

by

$$R_{\mathrm{stoch}}[f, t] := c(x_t, y_t, f(x_t)) + \lambda \Omega[f] \qquad (3)$$

and then performing gradient descent with respect to $R_{\mathrm{stoch}}[f, t]$. Here $t$ is either randomly chosen from $\{1, \ldots m\}$ or it is the new training instance observed at time $t$. Consequently the gradient of $R_{\mathrm{stoch}}[f, t]$ with respect to $f$ is

$$\partial_f R_{\mathrm{stoch}}[f, t] = c'(x_t, y_t, f(x_t))k(x_t, \cdot) + \lambda \partial_f \Omega[f] = c'(x_t, y_t, f(x_t))k(x_t, \cdot) + \lambda f. \quad (4)$$

The last equality holds if $\Omega[f] = \frac{1}{2}\|f\|^2$. Analogous results hold for general $\Omega[f] = \omega(\|f\|)$. The the update equations are hence straightforward:

$$f \rightarrow f - \Lambda \partial_f R_{\mathrm{stoch}}[f, t]. \qquad (5)$$

Here $\Lambda \in \mathbb{R}^+$ is the learning rate controlling the size of updates undertaken at each iteration. We will return to the issue of adjusting $(\lambda, \Lambda)$ at a later stage.

**Descent Algorithm**   For simplicity, assume that $\Omega[f] = \frac{1}{2}\|f\|^2$. In this case (5) becomes

$$f \rightarrow f - \Lambda \left( c'(x_t, y_t, f(x_t))k(x_t, \cdot) + \lambda f \right) = (1 - \lambda\Lambda)f - \Lambda c'(x_t, y_t, f(x_t))k(x_t, \cdot). \quad (6)$$

While (6) is convenient to use for a theoretical analysis, it is not directly amenable to computation. For this purpose we have to express $f$ as a kernel expansion

$$f(x) = \sum_i \alpha_i k(x_i, x) \qquad (7)$$

where the $x_i$ are (previously seen) training patterns. Then (6) becomes

$$
\begin{aligned}
\alpha_t \quad &\rightarrow \quad (1 - \lambda\Lambda)\alpha_t - \Lambda c'(x_t, y_t, f(x_t)) && (8) \\
&= \quad -\Lambda c'(x_t, y_t, f(x_t)) && \text{for } \alpha_t = 0 && (9) \\
\alpha_i \quad &\rightarrow \quad (1 - \lambda\Lambda)\alpha_i && \text{for } i \neq t. && (10)
\end{aligned}
$$

Eq. (8) means that at each iteration the kernel expansion may grow by one term. Furthermore, the cost for training at each step is not larger than the prediction cost: once we have computed $f(x_t)$, $\alpha_t$ is obtained by the value of the derivative of $c$ at $(x_t, y_t, f(x_t))$.

Instead of updating all coefficients $\alpha_i$ we may simply cache the power series $1, (1 - \lambda\Lambda), (1 - \lambda\Lambda)^2, (1 - \lambda\Lambda)^3, \ldots$ and pick suitable terms as needed. This is particularly useful if the derivatives of the loss function $c$ will only assume discrete values, say $\{-1, 0, 1\}$ as is the case when using the soft-margin type loss functions (see Section 3).

**Truncation**   The problem with (8) and (10) is that without any further measures, the number of basis functions $n$ will grow without bound. This is not desirable since $n$ determines the amount of computation needed for prediction. The regularization term helps us here. At each iteration the coefficients $\alpha_i$ with $i \neq t$ are shrunk by $(1 - \lambda\Lambda)$. Thus after $\tau$ iterations the coefficient $\alpha_i$ will be reduced to $(1 - \lambda\Lambda)^\tau \alpha_i$. Hence:

**Proposition 1 (Truncation Error)**   *For a loss function $c(x, y, f(x))$ with its first derivative bounded by $C$ and a kernel $k$ with bounded norm $\|k(x, \cdot)\| \leq X$, the truncation error in $f$ incurred by dropping terms $\alpha_i$ from the kernel expansion of $f$ after $\tau$ update steps is bounded by $\Lambda(1 - \lambda\Lambda)^\tau CX$. Furthermore, the total truncation error by dropping all terms which are at least $\tau$ steps old is bounded by*

$$\|f - f_{\mathrm{trunc}}\| \leq \sum_{i=1}^{t-\tau} \Lambda(1 - \lambda\Lambda)^{t-i}CX < \lambda^{-1}(1 - \lambda\Lambda)^\tau CX \qquad (11)$$

Here $f_{\text{trunc}} = \sum_{i=t-\tau+1}^{t} \alpha_i k(x_i, \cdot)$. Obviously the approximation quality increases exponentially with the number of terms retained.

The regularization parameter $\lambda$ can thus be used to control the storage requirements for the expansion. In addition, it naturally allows for distributions $P(x, y)$ that change over time in which cases it is desirable to *forget* instances $(x_i, y_i)$ that are much older than the average time scale of the distribution change [11].

## 3 Applications

We now proceed to applications of (8) and (10) to specific learning situations. We utilize the standard addition of the constant offset $b$ to the function expansion, i.e. $g(x) = f(x) + b$ where $f \in \mathcal{H}$ and $b \in \mathbb{R}$. Hence we also update $b$ into $b - \Lambda \partial_b R_{\text{stoch}}[g]$.

**Classification** A typical loss function in SVMs is the **soft margin**, given by $c(x, y, g(x)) = \max(0, 1 - yg(x))$. In this situation the update equations become

$$(\alpha_i, \alpha_t, b) \to \begin{cases} ((1 - \Lambda\lambda)\alpha_i, y_i\Lambda, b + \Lambda y_i) & \text{if } yg(x_t) < 1 \\ ((1 - \Lambda\lambda)\alpha_i, 0, b) & \text{otherwise.} \end{cases} \tag{12}$$

In **classification with the $\nu$-trick** we avoid having to fix the margin $\rho$ by treating it as a variable [15]. The value of $\rho$ is found automatically by using the loss function

$$c(x, y, g(x)) = \max(0, \rho - yg(x)) - \nu\rho \tag{13}$$

where $0 \leq \nu \leq 1$ is another parameter. Since $\nu$ has a much clearer intuitive meaning than $\rho$, it is easier to tune. On the other hand, one can show [15] that the specific choice of $\lambda$ has no influence on the estimate in $\nu$-SV classification. Therefore we may set $\lambda = 1$ and obtain

$$(\alpha_i, \alpha_t, b, \rho) \to \begin{cases} ((1 - \Lambda)\alpha_i, y_i\Lambda, b + \Lambda y_i, \rho + \Lambda(1 - \nu)) & \text{if } yg(x_t) < \rho \\ ((1 - \Lambda)\alpha_i, 0, b, \rho - \Lambda\nu) & \text{otherwise.} \end{cases} \tag{14}$$

Finally, if we choose the **hinge-loss**, $c(x, y, g(x)) = \max(0, -yg(x))$,

$$(\alpha_i, \alpha_t, b) \to \begin{cases} ((1 - \Lambda\lambda)\alpha_i, y_i\Lambda, b + \Lambda y_i) & \text{if } yg(x_t) < 0 \\ ((1 - \Lambda\lambda)\alpha_i, 0, b) & \text{otherwise.} \end{cases} \tag{15}$$

Setting $\lambda = 0$ recovers the kernel-perceptron algorithm. For nonzero $\lambda$ we obtain the kernel-perceptron with regularization.

**Novelty Detection** The results for novelty detection [14] are similar in spirit. The $\nu$-**setting** is most useful here particularly where the estimator acts as a warning device (e.g. network intrusion detection) and we would like to specify an upper limit on the frequency of alerts $f(x) < \rho$. The relevant loss function is $c(x, y, f(x)) = \max(0, \rho - f(x)) - \nu\rho$ and usually [14] one uses $f \in \mathcal{H}$ rather than $f + b$ where $b \in \mathbb{R}$ in order to avoid trivial solutions. The update equations are

$$(\alpha_i, \alpha_t, \rho) \to \begin{cases} ((1 - \Lambda)\alpha_i, \Lambda, \rho + \Lambda(1 - \nu)) & \text{if } f(x_t) < \rho \\ ((1 - \Lambda)\alpha_i, 0, \rho - \Lambda\nu) & \text{otherwise.} \end{cases} \tag{16}$$

Considering the update of $\rho$ we can see that on average only a fraction of $\nu$ observations will be considered for updates. Thus we only have to store a small fraction of the $x_i$.

**Regression** We consider the following four settings: squared loss, the $\varepsilon$-insensitive loss using the $\nu$-trick, Huber's robust loss function, and trimmed mean estimators. For convenience we will only use estimates $f \in \mathcal{H}$ rather than $g = f + b$ where $b \in \mathbb{R}$. The extension to the latter case is straightforward. We begin with **squared loss** where $c$ is given by $c(x, y, f(x)) = \frac{1}{2}(y - f(x))^2$. Consequently the update equation is

$$(\alpha_i, \alpha_t) \to ((1 - \lambda\Lambda)\alpha_i, \Lambda(y_t - f(x_t))). \tag{17}$$

This means that we have to store *every* observation we make, or more precisely, the prediction error we made on the observation. The $\varepsilon$-**insensitive loss** $c(x, y, f(x)) = \max(0, |y - f(x)| - \varepsilon)$ avoids this problem but introduces a new parameter in turn — the width of the insensitivity zone $\varepsilon$. By making $\varepsilon$ a variable of the optimization problem we have $c(x, y, f(x)) = \max(0, |y - f(x)| - \varepsilon) + \nu\varepsilon$. The update equations now have to be stated in terms of $\alpha_i, \alpha_t$, and $\varepsilon$ which is allowed to change during the optimization process. This leads to

$$(\alpha_i, \alpha_t, \varepsilon) \to \begin{cases} ((1 - \lambda\Lambda)\alpha_i, \Lambda\,\mathrm{sgn}(y_t - f(x_t)), \varepsilon + (1 - \nu)\Lambda) & \text{if } |y_t - f(x_t)| > \varepsilon \\ ((1 - \lambda\Lambda)\alpha_i, 0, \varepsilon - \Lambda\nu) & \text{otherwise.} \end{cases} \tag{18}$$

This means that every time the prediction error exceeds $\varepsilon$, we increase the insensitivity zone by $\Lambda\nu$. Likewise, if it is smaller than $\varepsilon$, the insensitive zone is decreased by $\Lambda(1 - \nu)$. Next let us analyze the case of regression with **Huber's robust loss**. The loss is given by

$$c(x, y, f(x)) = \begin{cases} |y - f(x)| - \frac{1}{2}\sigma & \text{if } |y - f(x)| \geq \sigma \\ \frac{1}{2\sigma}(y - f(x))^2 & \text{otherwise.} \end{cases} \tag{19}$$

As before we obtain update equations by computing the derivative of $c$ with respect to $f(x)$.

$$(\alpha_i, \alpha_t) \to \begin{cases} ((1 - \Lambda)\alpha_i, \Lambda\,\mathrm{sgn}(y_t - f(x_t))) & \text{if } |y_t - f(x_t)| > \sigma \\ ((1 - \Lambda)\alpha_i, \sigma^{-1}(y_t - f(x_t))) & \text{otherwise.} \end{cases} \tag{20}$$

Comparing (20) with (18) leads to the question whether $\sigma$ might not also be **adjusted adaptively**. This is a desirable goal since we may not know the amount of noise present in the data. While the $\nu$-setting allowed us to form such adaptive estimators for batch learning with the $\varepsilon$-insensitive loss, this goal has proven elusive for other estimators in the standard batch setting. In the online situation, however, such an extension is quite natural (see also [4]). All we need to do is make $\sigma$ a variable of the optimization problem and set

$$(\alpha_i, \alpha_t, \sigma) \to \begin{cases} ((1 - \Lambda)\alpha_i, \Lambda\,\mathrm{sgn}(y_t - f(x_t)), \sigma + \Lambda(1 - \nu)) & \text{if } |y_t - f(x_t)| > \sigma \\ ((1 - \Lambda)\alpha_i, \sigma^{-1}(y_t - f(x_t)), \sigma - \Lambda\nu) & \text{otherwise.} \end{cases} \tag{21}$$

## 4 Theoretical Analysis

Consider now the classification problem with the soft margin loss $c(x, y, f(x)) = \max(0, \rho - yf(x))$; here $\rho$ is a fixed margin parameter. Let $f_t$ denote the hypothesis of the online algorithm after seeing the first $t - 1$ observations. Thus, at time $t$, the algorithm receives an input $x_t$, makes its prediction $f_t(x_t)$, receives the correct outcome $y_t$, and updates its hypothesis into $f_{t+1}$ according to (5). We now wish to bound the cumulative risk $\sum_{t=1}^{m} R_{\mathrm{stoch}}[f_t, t]$. The motivation for such bounds is roughly as follows. Assume there is some fixed distribution $P$ from which the examples $(x_t, y_t)$ are drawn, and define

$$R_{\mathrm{P}}[f] := E_{(x,y)\sim P}[c(x, y, f(x))] + \lambda\Omega[f] \ .$$

Then it would be desirable for the online hypothesis $f_t$ to converge towards $f_* = \arg\min_f R_P[f]$. If we can show that the cumulative risk is asymptotically $mR_{\mathrm{stoch}}[f_*, t] + o(m)$, we see that at least in some sense $f_t$ does converge to $f_*$.

Hence, as a first step in our convergence analysis, we obtain an upper bound for the cumulative risk. In all the bounds of this section we assume $\Omega(f) = \frac{1}{2}\|f\|^2$.

**Theorem 1** *Let $((x_t, y_t))_{t=1}^m$ be an example sequence such that $k(x_t, x_t) \leq X^2$ for all $t$. Fix $B > 0$, and choose the learning rate $\Lambda = B/(Xm^{1/2})$. Then for any $g$ such that $\|g\| \leq B$ we have*

$$\sum_{t=1}^m R_{\text{stoch}}[f_t, t] \leq \sum_{t=1}^m R_{\text{stoch}}[g, t] + BXm^{1/2} + O(1) \ . \tag{22}$$

Notice that the bound does not depend on any probabilistic assumptions. If the example sequence is such that some fixed predictor $g$ has a small cumulative risk, then the cumulative risk of the online algorithm will also be small. There is a slight catch here in that the learning rate $\Lambda$ must be chosen a priori, and the optimal setting depends on $m$. The longer the sequence of examples, the smaller learning rate we want. We can avoid this by using a learning rate that starts from a fairly large value and decreases as learning progresses. This leads to a bound similar to Theorem 1 but with somewhat worse constant coefficients.

**Theorem 2** *Let $((x_t, y_t))_{t=1}^m$ be an example sequence such that $k(x_t, x_t) \leq X^2$ for all $t$. Fix $B > 0$, and use at update $t$ the learning rate $\Lambda_t = 1/(3\lambda t^{1/2})$. Then for any $g$ such that $\|g\| \leq B$ we have*

$$\sum_{t=1}^m R_{\text{stoch}}[f_t, t] \leq \sum_{t=1}^m R_{\text{stoch}}[g, t] + 2\lambda(B + X/\lambda)^2 m^{1/2} + O(1) \ . \tag{23}$$

Let us now consider the implications of Theorem 2 to a situation in which we assume that the examples $(x_t, y_t)$ are i.i.d. according to some fixed distribution $P$.

**Theorem 3** *Let $P$ be a distribution over $\mathcal{X} \times \mathcal{Y}$, such that $k(x, x) \leq X^2$ holds with probability $1$ for $(x, y) \sim P$. Let $\hat{f}_m = (1/m) \sum_{t=1}^{m-1} f_t$ where $f_t$ is the $t$-th online hypothesis based on an example sequence $((x_t, y_t))_{t=1}^m$ that is drawn i.i.d. according to $P$. Fix $B > 0$, and use at update $t$ the learning rate $\Lambda_t = 1/(3\lambda t^{1/2})$. Then for any $g$ such that $\|g\| \leq B$ we have*

$$E[R_{\text{P}}[\hat{f}_m]] \leq R_{\text{P}}[g] + 2\lambda(B + X/\lambda)^2 m^{-1/2} + O(m^{-1}) \ . \tag{24}$$

If we know in advance how many examples we are going to draw, we can use a fixed learning rate as in Theorem 1 and obtain somewhat better constants.

## 5   Experiments and Discussion

In our experiments we studied the performance of online $\nu$-SVM algorithms in various settings. They always yielded competitive performance. Due to space constraints we only report the findings in novelty detection as given in Figure 1 (where the training algorithm was fed the patterns sans class labels).

Already after one pass through the USPS database (5000 training patterns, 2000 test patterns, each of them of size $16 \times 16$ pixels), which took in MATLAB less than 15s on a 433MHz Celeron, the results can be used for weeding out badly written digits. The $\nu$-setting was used (with $\nu = 0.01$) to allow for a fixed fraction of detected "outliers." Based on the theoretical analysis of Section 4 we used a decreasing learning rate with $\lambda \propto t^{-\frac{1}{2}}$.

**Conclusion**   We have presented a range of simple online kernel-based algorithms for a variety of standard machine learning tasks. The algorithms have constant memory requirements and are computationally cheap at each update step. They allow the ready application of powerful kernel based methods such as novelty detection to online and time-varying problems.

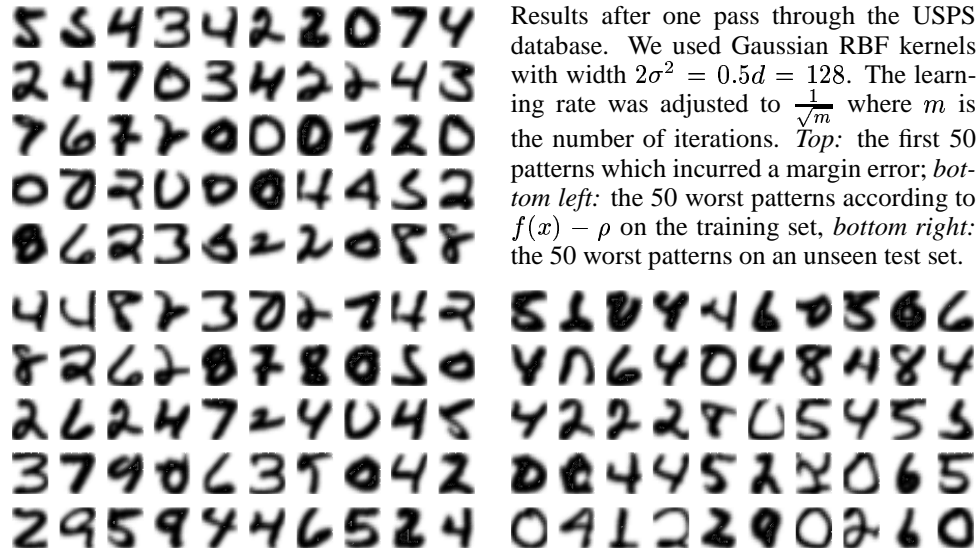

Results after one pass through the USPS database. We used Gaussian RBF kernels with width $2\sigma^2 = 0.5d = 128$. The learning rate was adjusted to $\frac{1}{\sqrt{m}}$ where $m$ is the number of iterations. *Top:* the first 50 patterns which incurred a margin error; *bottom left:* the 50 worst patterns according to $f(x) - \rho$ on the training set, *bottom right:* the 50 worst patterns on an unseen test set.

Figure 1: Online novelty detection on the USPS dataset with $\nu = 0.01$.

**Acknowledgments** A.S. was supported by the DFG under grant Sm 62/1-1, J.K. & R.C.W. were supported by the ARC. The authors thank Paul Wankadia for help with the implementation.

# References

[1] K. P. Bennett and O. L. Mangasarian. Robust linear programming discrimination of two linearly inseparable sets. *Optimization Methods and Software*, 1:23–34, 1992.

[2] G. Cauwenberghs and T. Poggio. Incremental and decremental support vector machine learning. In T. K. Leen, T. G. Dietterich, and V. Tresp, editors, *Advances in Neural Information Processing Systems 13*, pages 409–415. MIT Press, 2001.

[3] L. Csató and M. Opper. Sparse representation for gaussian process models. In T. K. Leen, T. G. Dietterich, and V. Tresp, editors, *Advances in Neural Information Processing Systems 13*, pages 444–450. MIT Press, 2001.

[4] J. Friedman, T. Hastie, and R. Tibshirani. Additive logistic regression: a statistical view of boosting. Technical report, Stanford University, Dept. of Statistics, 1998.

[5] C. Gentile. A new approximate maximal margin classification algorithm. In T. K. Leen, T. G. Dietterich, and V. Tresp, editors, *Advances in Neural Information Processing Systems 13*, pages 500–506. MIT Press, 2001.

[6] T. Graepel, R. Herbrich, and R. C. Williamson. From margin to sparsity. In T. K. Leen, T. G. Dietterich, and V. Tresp, editors, *Advances in Neural Information Processing Systems 13*, pages 210–216. MIT Press, 2001.

[7] Y. Guo, P. Bartlett, A. Smola, and R. C. Williamson. Norm-based regularization of boosting. Submitted to *Journal of Machine Learning Research*, 2001.

[8] M. Herbster. Learning additive models online with fast evaluating kernels. In *Proc. 14th Annual Conference on Computational Learning Theory (COLT)*, pages 444–460. Springer, 2001.

[9] P. J. Huber. Robust statistics: a review. *Annals of Statistics*, 43:1041, 1972.

[10] G. S. Kimeldorf and G. Wahba. Some results on Tchebycheffian spline functions. *J. Math. Anal. Applic.*, 33:82–95, 1971.

[11] J. Kivinen, A.J. Smola, and R.C. Williamson. Large margin classification for moving targets. Unpublished manuscript, 2001.

[12] Y. Li and P.M. Long. The relaxed online maximum margin algorithm. In S. A. Solla, T. K. Leen, and K.-R. Müller, editors, *Advances in Neural Information Processing Systems 12*, pages 498–504. MIT Press, 1999.

[13] L. Mason, J. Baxter, P. L. Bartlett, and M. Frean. Functional gradient techniques for combining hypotheses. In A. J. Smola, P. L. Bartlett, B. Schölkopf, and D. Schuurmans, editors, *Advances in Large Margin Classifiers*, Cambridge, MA, 2000. MIT Press. 221–246.

[14] B. Schölkopf, J. Platt, J. Shawe-Taylor, A. J. Smola, and R. C. Williamson. Estimating the support of a high-dimensional distribution. *Neural Computation*, 13(7), 2001.

[15] B. Schölkopf, A. Smola, R. C. Williamson, and P. L. Bartlett. New support vector algorithms. *Neural Computation*, 12(5):1207–1245, 2000.

[16] V. Vapnik, S. Golowich, and A. Smola. Support vector method for function approximation, regression estimation, and signal processing. In M. Mozer, M. Jordan, and T. Petsche, editors, *Advances in Neural Information Processing Systems 9*, pages 281–287, Cambridge, MA, 1997. MIT Press.
